# Computing Gaussian Mixture Models with EM using Equivalence Constraints

**Noam Shental**
Computer Science & Eng.
Center for Neural Computation
Hebrew University of Jerusalem
Jerusalem, Israel 91904
fenoam@cs.huji.ac.il

**Aharon Bar-Hillel**
Computer Science & Eng.
Center for Neural Computation
Hebrew University of Jerusalem
Jerusalem, Israel 91904
aharonbh@cs.huji.ac.il

**Tomer Hertz**
Computer Science & Eng.
Center for Neural Computation
Hebrew University of Jerusalem
Jerusalem, Israel 91904
tomboy@cs.huji.ac.il

**Daphna Weinshall**
Computer Science & Eng.
Center for Neural Computation
Hebrew University of Jerusalem
Jerusalem, Israel 91904
daphna@cs.huji.ac.il

## Abstract

Density estimation with Gaussian Mixture Models is a popular generative technique used also for clustering. We develop a framework to incorporate side information in the form of *equivalence constraints* into the model estimation procedure. *Equivalence constraints* are defined on pairs of data points, indicating whether the points arise from the same source (positive constraints) or from different sources (negative constraints). Such constraints can be gathered automatically in some learning problems, and are a natural form of supervision in others. For the estimation of model parameters we present a closed form EM procedure which handles positive constraints, and a Generalized EM procedure using a Markov net which handles negative constraints. Using publicly available data sets we demonstrate that such side information can lead to considerable improvement in clustering tasks, and that our algorithm is preferable to two other suggested methods using the same type of side information.

## 1 Introduction

We are used to thinking about learning from labels as supervised learning, and learning without labels as unsupervised learning, where 'supervised' implies the need for human intervention. However, in unsupervised learning we are not limited to using data statistics only. Similarly supervised learning is not limited to using labels. In this work we focus on semi-supervised learning using side-information, which is *not* given as labels. More specifically, we use unlabeled data augmented by *equivalence constraints* between pairs of data points, where the constraints determine whether each pair was generated by the

same source or by different sources. Such constraints may be acquired without human intervention in a broad class of problems, and are a natural form of supervision in other scenarios. We show how to incorporate *equivalence constraints* into the EM algorithm [1], in order to fit a generative Gaussian mixture model to the data.

Density estimation with Gaussian mixture models is a popular generative technique, mostly because it is computationally tractable and often produces good results. However, even when the approach is successful, the underlying assumptions (i.e., that the data is generated by a mixture of Gaussian sources) may not be easily justified. It is therefore important to have additional information which can steer the GMM estimation in the "right" direction. In this paper we propose to incorporate equivalence constraints into an EM parameter estimation algorithm. One added value may be a faster convergence to a high likelihood solution. Much more importantly, the constraints change the GMM likelihood function and therefore may lead the estimation procedure to choose a better solution which would have otherwise been rejected (due to low relative likelihood in the unconstrained GMM density model). Ideally the solution obtained with side information will be more faithful to the desired results. A simple example demonstrating this point is shown in Fig. 1.

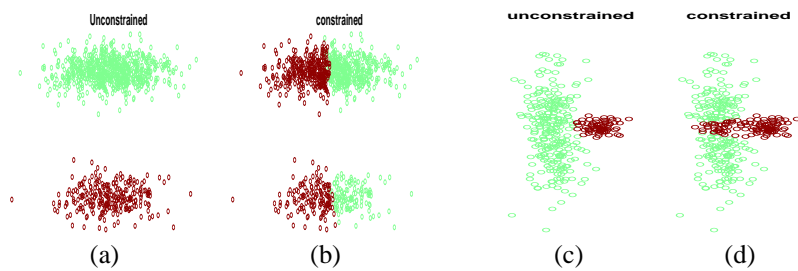

Figure 1: Illustrative examples for the importance of *equivalence constraints*. Left: the data set consists of 2 *vertically aligned* classes - (a) given no additional information, the EM algorithm identifies two *horizontal* classes, and this can be shown to be the maximum likelihood solution (with log likelihood of $-3500$ vs. log likelihood of $-2800$ for the solution in (b)); (b) additional side information in the form of equivalence constraints changes the probability function and we get a vertical partition as the most likely solution. Right: the dataset consists of two classes with partial overlap - (c) without constraints the most likely solution includes two *non*-overlapping sources; (d) with constraints the correct model with overlapping classes was retrieved as the most likely solution. In all plots only the class assignment of novel *un-constrained* points is shown.

*Equivalence constraints* are binary functions of pairs of points, indicating whether the two points come from the same source or from two different sources. We denote the first case as "is-equivalent" constraints, and the second as "not-equivalent" constraints. As it turns out, "is-equivalent" constraints can be easily incorporated into EM, while "not-equivalent" constraints require heavy duty inference machinery such as Markov networks. We describe the derivations in Section 2.

Our choice to use equivalence constraints is motivated by the relative abundance of *equivalence constraints* in real life applications. In a broad family of applications, *equivalence constraints* can be obtained without supervision. Maybe the most important source of unsupervised *equivalence constraints* is temporal continuity in data; for example, in video indexing a sequence of faces obtained from successive frames in roughly the same location are likely to contain the same unknown individual. Furthermore, there are several learning applications in which *equivalence constraints* are the natural form of supervision.

One such scenario occurs when we wish to enhance a retrieval engine using supervision provided by its users. The users may be asked to help annotate the retrieved set of data points, in what may be viewed as 'generalized relevance feedback'. The categories given

by the users have subjective names that may be inconsistent. Therefore, we can only extract *equivalence constraints* from the feedback provided by the users. Similar things happen in a 'distributed learning' scenario, where supervision is provided by several uncoordinated teachers. In such scenarios, when *equivalence constraints* are obtained in a supervised manner, our method can be viewed as a semi-supervised learning technique. Most of the work in the field of semi-supervised learning focused on the case of partial labels augmenting a large unlabeled data set [4, 8, 5].

A few recent papers use side information in the form of *equivalence constraints* [6, 7, 10]. In [9] *equivalence constraints* were introduced into the K-means clustering algorithm. The algorithm is closely related to our work since it allows for the incorporation of both "is-equivalent" and "not-equivalent" constraints. In [3] equivalence constraints were introduced into the complete linkage clustering algorithm. In comparison with both approaches, we gain significantly better clustering results by introducing the constraints into the EM algorithm. One reason may be that the EM of a Gaussian mixture model is preferable as a clustering algorithm. More importantly, the probabilistic semantics of the EM procedure allows for the introduction of constraints in a principled way, thus overcoming many drawbacks of the heuristic approaches. Comparative results are given in Section 3, demonstrating the very significant advantage of our method over the two alternative constrained clustering algorithms using a number of data sets from the UCI repository and a large database of facial images [2].

## 2    Constrained EM: the update rules

A Gaussian mixture model (GMM) is a parametric statistical model which assumes that the data originates from a weighted sum of several Gaussian sources. More formally, a GMM is given by $p(x|\Theta) = \Sigma_{l=1}^{M} \alpha_l p(x|\theta_l)$, where $\alpha_l$ denotes the weight of each Gaussian, $\theta_l$ its respective parameters, and $M$ denotes the number of Gaussian sources in the GMM. EM is a widely used method for estimating the parameter set of the model ($\Theta$) using unlabeled data [1]. *Equivalence constraints* modify the 'E' (expectation computation) step, such that the sum is taken only over assignments which comply with the given constraints (instead of summing over *all* possible assignments of data points to sources).

It is important to note that there is a basic difference between "is-equivalent" (positive) and "not-equivalent" (negative) constraints: While positive constraints are transitive (i.e. a group of pairwise "is-equivalent" constraints can be merged using a transitive closure), negative constraints are not transitive. The outcome of this difference is expressed in the complexity of incorporating each type of constraint into the EM formulation. Therefore, we begin by presenting a formulation for positive constraints (Section 2.1), and then present a different formulation for negative constraints (Section 2.2). A unified formulation for both types of constraints immediately follows, and the details are therefore omitted.

### 2.1    Incorporating positive constraints

Let a *chunklet* denote a small subset of data points that are known to belong to a single unknown class. Chunklets may be obtained by applying the transitive closure to the set of "is-equivalent" constraints.
Assume as given a set of unlabeled data points and a set of chunklets. In order to write down the likelihood of a given assignment of points to sources, a probabilistic model of how chunklets are obtained must be specified. We consider two such models:

1. Chunklets are sampled i.i.d, with respect to the weight of their corresponding source (points within each chunklet are also sampled i.i.d).
2. Data points are sampled i.i.d, without any knowledge about their class membership, and only afterwards chunklets are selected from these points.

The first assumption may be appropriate when chunklets are automatically obtained using temporal continuity. The second sampling assumption is appropriate when *equivalence constraints* are obtained using *distributed learning*. When incorporating these sampling assumptions into the EM formulation for GMM fitting, different algorithms are obtained: Using the first assumption we obtain closed-form update rules for all of the GMM parameters. When the second sampling assumption is used there is no closed-form solution for the sources' weights. In this section we therefore restrict the discussion to the first sampling assumption only; the discussion of the second sampling assumption, where generalized EM must be used, is omitted.

More specifically, let $p(x) = \sum_{l=1}^{M} \alpha_l \, p_l(x|\theta_l)$ denote our GMM. Each $p_l(x|\theta_l)$ term is a Gaussian parameterized by $\theta_l = (\mu_l, \Sigma_l)$ with a mixing coefficient $\alpha_l$. Let $\mathbf{X}$ denote the set of all data points, $\mathbf{X} = \{x_i\}_{i=1}^{N}$. Let $\{X_j\}_{j=1}^{L}$, $L \leq N$ denote the distinct chunklets, where each $X_j$ is a set of points $x_i$ such that $\bigcup_{j=1}^{L} X_j = \{x_i\}_{i=1}^{N}$ (unconstrained data points appear as chunklets of size one). Let $\mathbf{Y} = \{y_i\}_{i=1}^{N}$ denote the source assignment of the respective data-points, and $Y_j = \{y_j^1 \dots y_j^{|X_j|}\}$ denote the source assignment of the chunklet $X_j$. Finally, let $E_\Omega$ denote the event $\{\mathbf{Y}$ complies with the constraints$\}$.

The expectation of the log likelihood is the following:

$$E[log(p(\mathbf{X},\mathbf{Y}|\Theta^{new}, E_\Omega))|\mathbf{X}\,\Theta^{old}, E_\Omega] = \sum_{\mathbf{Y}} log(p(\mathbf{X},\mathbf{Y}|\Theta^{new}, E_\Omega)) \cdot p(\mathbf{Y}|\mathbf{X},\Theta^{old}, E_\Omega) \quad (1)$$

where $\sum_{\mathbf{Y}}$ stands for a summation over all assignments of points to sources: $\sum_{\mathbf{Y}} \equiv \sum_{y_1=1}^{M} \cdots \sum_{y_N=1}^{M}$. In the following discussion we shall also reorder the sum according to chunklets: $\sum_{\mathbf{Y}} \equiv \sum_{Y_1} \cdots \sum_{Y_L}$, where $\sum_{Y_j}$ stands for $\sum_{y_1^j} \cdots \sum_{y_{|X_j|}^j}$.

First, using Bayes rule and the independence of chunklets, we can write

$$
\begin{aligned}
p(\mathbf{Y}|\mathbf{X},\Theta^{old}, E_\Omega) &= \frac{p(E_\Omega|\mathbf{Y},\mathbf{X},\Theta^{old})\, p(\mathbf{Y}|\mathbf{X},\Theta^{old})}{\sum_{\mathbf{Y}} p(E_\Omega|\mathbf{Y},\mathbf{X},\Theta^{old})\, p(\mathbf{Y}|\mathbf{X},\Theta^{old})} \\
&= \frac{\prod_{j=1}^{L} \delta_{Y_j}\, p(Y_j|X_j,\Theta^{old})}{\sum_{Y_1} \cdots \sum_{Y_L} \prod_{j=1}^{L} \delta_{Y_j}\, p(Y_j|X_j,\Theta^{old})}
\end{aligned} \quad (2)
$$

where $\delta_{Y_j} \equiv \delta_{y_1^j, \dots, y_{|X_j|}^j}$ equals 1 if all the points in chunklet $i$ have the same label, and 0 otherwise.

Next, using chunklet independence and the independence of points within a chunklet we see that

$$
\begin{aligned}
p(\mathbf{X},\mathbf{Y}|\Theta^{new}, E_\Omega) &= p(\mathbf{Y}|\Theta^{new}, E_\Omega)\, p(\mathbf{X}|\mathbf{Y},\Theta^{new}, E_\Omega) \\
&= \prod_{j=1}^{L} \alpha_{y_j} \prod_{i=1}^{N} p(x_i|y_i, \Theta^{new})
\end{aligned}
$$

Hence the log-likelihood is:

$$log\, p(\mathbf{X},\mathbf{Y}|\Theta^{new}, E_\Omega) = \sum_{j=1}^{L} \sum_{x_i \in X_j} log\, p(x_i|y_i, \Theta^{new}) + \sum_{j=1}^{L} log(\alpha_{y_j}) \quad (3)$$

Finally, we substitute (3) and (2) into (1); after some manipulations, we obtain the following expression:

$$
\begin{aligned}
E(LogLikelihood) &= \sum_{l=1}^{M} \sum_{j=1}^{L} \sum_{x_i \in X_j} log\, p(x_i|l, \Theta^{new}) \cdot p(Y_j = l|X_j, \Theta^{old}) \\
&\quad + \sum_{l=1}^{M} \sum_{j=1}^{L} log\, \alpha_l \cdot p(Y_j = l|X_j, \Theta^{old})
\end{aligned}
$$

where the chunklet posterior probability is:

$$p(Y_j = l | X_j, \Theta^{old}) = \frac{\alpha_l^{old} \prod_{x_i \in X_j} p(x_i | y_i^j = l, \Theta^{old})}{\sum_{m=1}^{M} \alpha_m^{old} \prod_{x_i \in X_j} p(x_i | y_i^j = m, \Theta^{old})}$$

To find the update rule for each parameter, we differentiate (4) with respect to $\mu_l$, $\Sigma_l$ and $\alpha_l$. We get the following rules:

$$\alpha_l^{new} = \frac{1}{L} \sum_{j=1}^{L} p(Y_j = l | X_j, \Theta^{old})$$

$$\mu_l^{new} = \frac{\sum_{j=1}^{L} \bar{X}_j p(Y_j = l | X_j, \Theta^{old}) |X_j|}{\sum_{j=1}^{L} p(Y_j = l | X_j, \Theta^{old}) |X_j|}$$

$$\Sigma_l^{new} = \frac{\sum_{j=1}^{L} \Sigma_{jl}^{new} p(Y_j = l | X_j, \Theta^{old}) |X_j|}{\sum_{j=1}^{L} p(Y_j = l | X_j, \Theta^{old}) |X_j|}$$

where $\bar{X}_j$ denotes the sample mean of the points in chunklet $j$, $|X_j|$ denotes the number of points in chunklet $j$, and $\Sigma_{jl}^{new}$ denotes the sample covariance matrix of the $j$th chunklet of the $l$th class.

As can be readily seen, the update rules above effectively treat each chunklet as a single data point weighed according to the number of elements in it.

## 2.2 Incorporating negative constraints

The probabilistic description of a data set using a GMM attaches to each data point two random variables: an observable and a hidden. The hidden variable of a point describes its source label, while the data point itself is an observed example from the source. Each pair of observable and hidden variables is assumed to be independent of the other pairs. However, negative *equivalence constraints* violate this assumption, as dependencies between the hidden variables are introduced.

Specifically, assume we have a group $\Omega = \{(a_i^1, a_i^2)\}_{i=1}^{P}$ of index pairs corresponding to $P$ pairs of points that are negatively constrained, and define the event $E_\Omega = \{\mathbf{Y} \text{ complies with the constraints}\}$. Now

$$p(\mathbf{X}, \mathbf{Y} | \Theta, E_\Omega) = p(\mathbf{X} | \mathbf{Y}, \Theta, E_\Omega) \, p(\mathbf{Y} | \Theta, E_\Omega) = \frac{p(\mathbf{X} | \mathbf{Y}, \Theta) \, p(E_\Omega | \mathbf{Y}) \, p(\mathbf{Y} | \Theta)}{p(E_\Omega | \Theta)}$$

Let $Z$ denote the constant $p(E_\Omega | \Theta)$. Assuming sample independence, it follows that $p(\mathbf{X} | \mathbf{Y}, \Theta) \cdot p(\mathbf{Y} | \Theta) = \prod_{i=1}^{N} p(y_i | \Theta) p(x_i | y_i, \Theta)$. By definition $p(E_\Omega | \mathbf{Y}) = 1_{\mathbf{Y} \in E_\Omega}$, hence

$$p(\mathbf{X}, \mathbf{Y} | \Theta, E_\Omega) = \frac{1}{Z} 1_{\mathbf{Y} \in E_\Omega} \prod_{i=1}^{N} p(y_i | \Theta) p(x_i | y_i, \Theta) \tag{4}$$

Expanding $1_{\mathbf{Y} \in E_\Omega}$ gives the following expression

$$p(\mathbf{X}, \mathbf{Y} | \Theta, E_\Omega) = \frac{1}{Z} \prod_{(a_i^1, a_i^2)} (1 - \delta_{y_{a_i^1}, y_{a_i^2}}) \prod_{i=1}^{N} p(y_i | \Theta) p(x_i | y_i, \Theta) \tag{5}$$

As a product of local components, the distribution in (5) can be readily described using a Markov network. The network nodes are the hidden source variables and the observable

data point variables. The potential $p(x_i|y_i, \Theta)$ connects each observable data point, in a Gaussian manner, to a hidden variable corresponding to the label of its source. Each hidden source node holds an initial potential of $p(y_i|\Theta)$ reflecting the prior of the cluster weights. Negative constraints are expressed by edges between hidden variables which prevent them from having the same value. A potential over an edge $(a_i^1 - a_i^2)$ is expressed by $1 - \delta_{y_{a_i^1}, y_{a_i^2}}$ (see Fig. 2).

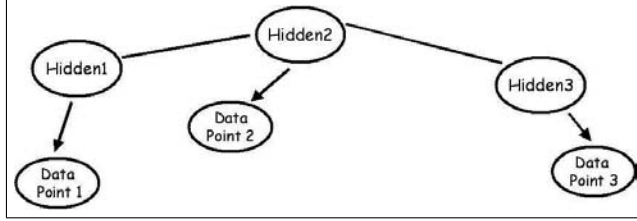

Figure 2: An illustration of the Markov network required for incorporating "not-equivalent" constraints. Data points 1 and 2 have a negative constraint, and so do points 2 and 3.

We derived an EM procedure which maximizes $log(p(\mathbf{X}|\Theta, E_\Omega))$ entailed by this distribution. The update rules for $\mu_l$ and $\Sigma_l$ are still

$$\mu_l^{new} = \frac{\sum_{i=1}^N x_i p(y_i = l|\mathbf{X}, \Theta^{old}, E_\Omega)}{\sum_{i=1}^N p(y_i = l|\mathbf{X}, \Theta^{old}, E_\Omega)}, \quad \Sigma_l^{new} = \frac{\sum_{i=1}^N \widehat{\Sigma_i l} \, p(y_i = l|\mathbf{X}, \Theta^{old}, E_\Omega)}{\sum_{i=1}^N p(y_i = l|\mathbf{X}, \Theta^{old}, E_\Omega)}$$

where $\widehat{\Sigma_i l} = (x_i - \mu_l^{new})(x_i - \mu_l^{new})^T$ denotes the sample covariance matrix. Note, however, that now the vector of probabilities $p(y_i = l|\mathbf{X}, \Theta^{old}, E_\Omega)$ is inferred using the net.

The update rule of $\alpha_l = p(y_i = l|\Theta_{new}, E_\Omega)$ is more intricate, since this parameter appears in the normalization factor $Z$ in the likelihood expression (4):

$$Z = p(E_\Omega|\Theta) = \sum_{\mathbf{Y}} p(\mathbf{Y}|\Theta) p(E_\Omega|\mathbf{Y}) = \sum_{y_1} ... \sum_{y_N} \prod_{i=1}^N \alpha_{y_i} \prod_{(a_i^1, a_i^2)} (1 - \delta_{y_{a_i^1}, y_{a_i^2}}) \quad (6)$$

This factor can be calculated using a net which is similar to the one discussed above but lacks the observable nodes. We use such a net to calculate $Z$ and differentiate it w.r.t $\alpha_l$, after which we perform gradient ascent. Alternatively, we can approximate $Z$ by assuming that the pairs of negatively constrained points are disjoint. Using such an assumption, $Z$ is reduced to the relatively simple expression: $Z = (1 - \sum_{i=1}^M \alpha_i^2)^P$. This expression for $Z$ can be easily differentiated, and can be used in the Generalized EM scheme. Although the assumption is not valid in most cases, it is a reasonable approximation in sparse networks, and our empirical tests show that it gives good results.

## 3 Experimental results

In order to evaluate the performance of our EM derivations and compare it to the constrained K-means [9] and constrained complete linkage [3] algorithms, we tested all 3 algorithms using several data sets from the UCI repository and a real multi-class facial image database [2]. We simulated a 'distributed learning' scenario in order to obtain side information. In this scenario *equivalence constraints* are obtained by employing $N$ uncoordinated teachers. Each teacher is given a random selection of $K$ data points from the data set, and is then asked to partition this set of points into equivalence classes. The constraints provided

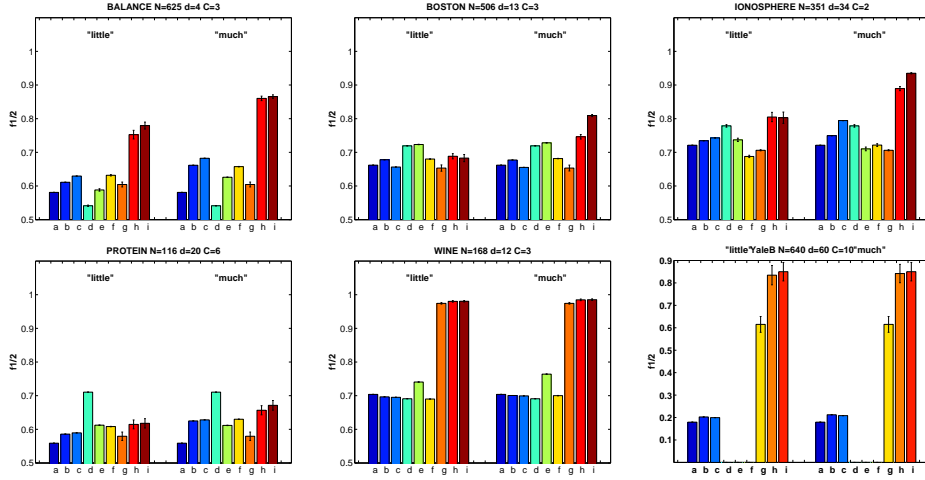

Figure 3: Combined precision and recall scores ($f_{\frac{1}{2}}$) of several clustering algorithms over 5 data sets from the UCI repository, and 1 facial image database (YaleB). The YaleB dataset contained a total of 640 images including 64 frontal pose images of 10 different subjects. In this dataset the variability between images of the same person was due mainly to different lighting conditions. Results are presented for the following algorithms: (a) K-means, (b) constrained K-means using only positive constraints, (c) constrained K-means using both positive and negative constraints, (d) complete linkage, (e) complete linkage using positive constraints, (f) complete linkage using both positive and negative constraints, (g) regular EM, (h) EM using positive constraints, and (i) EM using both positive and negative constraints. In each panel results are shown for two cases, using 15% of the data points in constraints (left bars) and 30% of the points constrained (right bars). The results were averaged over 100 realizations of constraints for the UCI datasets, and 1000 realizations for the YaleB dataset. Also shown are the names of the data sets used and some of their parameters: N - the size of the data set; C - the number of classes; d - the dimensionality of the data.

by the teachers are gathered and used as *equivalence constraints*. Each of the 3 algorithms (constrained EM, constrained K-means, and constrained complete linkage) was tested in three modes: (i) basic algorithm without using any side information, (ii) constrained version using only positive *equivalence constraints*, and (iii) constrained version using both positive and negative *equivalence constraints*. The results of the 9 algorithmic variants are compared in Fig. 3.

In the simulations, the number of constrained points was determined by the number of teachers $N$ and the size of the subset $K$ given to each. By controlling the product $NK$ we controlled the amount of side information provided to the learning algorithms. We experimented with two conditions: using "little" side information (approximately 15% of the data points are constrained) and using "much" side information (approximately 30% of the points are constrained). All algorithms were given initial conditions that did not take into account the available *equivalence constraints*. The results were evaluated using a combined measure of precision $P$ and recall $R$ scores: $f_{\frac{1}{2}} = \frac{2PR}{R+P}$.

Several effects can clearly be seen in the results reported in Fig. 3:

- The constrained EM outperformed the two alternative algorithms in almost all cases, while showing substantial improvement over the baseline EM. The one case where constrained complete linkage outperformed all other algorithms involved the "wine" dataset. In this dataset the data lies in a high-dimensional space ($\mathcal{R}^{12}$) and therefore the number of model parameters to be estimated by the EM

algorithm is relatively large. The EM procedure was not able to fit the data well even with constraints, probably due to the fact that only 168 data points were available for training.

- Introducing side information in the form of *equivalence constraints* clearly improves the results of both K-means and the EM algorithms. This is not always true for the constrained complete linkage algorithm. As the amount of side-information increases, performance typically improves.

- Most of the improvement can be attributed to the positive constraints, and can be achieved using our closed form EM version. In most cases adding the negative constraints contributes a small but significant improvement over results obtained when using only positive constraints.

# References

[1] A. P. Dempster, N. M. Laird, and D. B. Rubin. Maximum likelihood from incomplete data via the EM algorithm. *JRSSB*, 39:1–38, 1977.

[2] A. Georghiades, P.N. Belhumeur, and D.J. Kriegman. From few to many: Generative models for recognition under variable pose and illumination. *IEEE international Conference on Automatic Face and Gesture Recognition*, pages 277–284, 2000.

[3] D. Klein, Sepandar D. Kamvar, and Christopher D. Manning. From instance-level constraints to space-level constraints: Making the most of prior knowledge in data clustering. In *ICML*, 2002.

[4] D. Miller and S. Uyar. A mixture of experts classifier with learning based on both labelled and unlabelled data. In M. C. Mozer, M. I. Jordan, and T. Petsche, editors, *NIPS 9*, pages 571–578. MIT Press, 1997.

[5] K. Nigam, A.K. McCallum, S. Thrun, and T.M. Mitchell. Learning to classify text from labeled and unlabeled documents. In *Proceedings of AAAI-98*, pages 792–799, Madison, US, 1998. AAAI Press, Menlo Park, US.

[6] P.J. Phillips. Support vector machines applied to face recognition. In M. C. Mozer, M. I. Jordan, and T. Petsche, editors, *NIPS 11*, page 803ff. MIT Press, 1998.

[7] N. Shental, T. Hertz, D. Weinshall, and M. Pavel. Adjustment learning and relevant component analysis. In A. Heyden, G. Sparr, M. Nielsen, and P. Johansen, editors, *Computer Vision - ECCV 2002*, volume 4, page 776ff, 2002.

[8] M. Szummer and T. Jaakkola. Partially labeled classification with markov random walks. In *NIPS*, volume 14. The MIT Press, 2001.

[9] K. Wagstaff, C. Cardie, S. Rogers, and S. Schroedl. Constrained K-means clustering with background knowledge. In *Proc. 18th International Conf. on Machine Learning*, pages 577–584. Morgan Kaufmann, San Francisco, CA, 2001.

[10] E.P Xing, A.Y. Ng, M.I. Jordan, and S. Russell. Distance metric learnign with application to clustering with side-information. In *Advances in Neural Information Processing Systems*, volume 15. The MIT Press, 2002.
